# Laplacian Score for Feature Selection

**Xiaofei He**[1]     **Deng Cai**[2]     **Partha Niyogi**[1]
[1] Department of Computer Science, University of Chicago
{xiaofei, niyogi}@cs.uchicago.edu
[2] Department of Computer Science, University of Illinois at Urbana-Champaign
dengcai2@uiuc.edu

## Abstract

In supervised learning scenarios, feature selection has been studied widely in the literature. Selecting features in unsupervised learning scenarios is a much harder problem, due to the absence of class labels that would guide the search for relevant information. And, almost all of previous unsupervised feature selection methods are "wrapper" techniques that require a learning algorithm to evaluate the candidate feature subsets. In this paper, we propose a "filter" method for feature selection which is independent of any learning algorithm. Our method can be performed in either supervised or unsupervised fashion. The proposed method is based on the observation that, in many real world classification problems, data from the same class are often close to each other. The importance of a feature is evaluated by its power of locality preserving, or, **Laplacian Score**. We compare our method with data variance (unsupervised) and Fisher score (supervised) on two data sets. Experimental results demonstrate the effectiveness and efficiency of our algorithm.

## 1   Introduction

Feature selection methods can be classified into "wrapper" methods and "filter" methods [4]. The wrapper model techniques evaluate the features using the learning algorithm that will ultimately be employed. Thus, they "wrap" the selection process around the learning algorithm. Most of the feature selection methods are wrapper methods. Algorithms based on the filter model examine intrinsic properties of the data to evaluate the features prior to the learning tasks. The filter based approaches almost always rely on the class labels, most commonly assessing correlations between features and the class label. In this paper, we are particularly interested in the filter methods. Some typical filter methods include data variance, Pearson correlation coefficients, Fisher score, and Kolmogorov-Smirnov test.

Most of the existing filter methods are supervised. Data variance might be the simplest *unsupervised* evaluation of the features. The variance along a dimension reflects its representative power. Data variance can be used as a criteria for feature selection and extraction. For example, Principal Component Analysis (PCA) is a classical feature extraction method which finds a set of mutually orthogonal basis functions that capture the directions of maximum variance in the data.

Although the data variance criteria finds features that are useful for representing data, there

is no reason to assume that these features must be useful for discriminating between data in different classes. Fisher score seeks features that are efficient for discrimination. It assigns the highest score to the feature on which the data points of different classes are far from each other while requiring data points of the same class to be close to each other. Fisher criterion can be also used for feature extraction, such as Linear Discriminant Analysis (LDA).

In this paper, we introduce a novel feature selection algorithm called **Laplacian Score** (LS). For each feature, its Laplacian score is computed to reflect its locality preserving power. LS is based on the observation that, two data points are probably related to the same topic if they are close to each other. In fact, in many learning problems such as classification, the local structure of the data space is more important than the global structure. In order to model the local geometric structure, we construct a nearest neighbor graph. LS seeks those features that respect this graph structure.

## 2 Laplacian Score

Laplacian Score (LS) is fundamentally based on Laplacian Eigenmaps [1] and Locality Preserving Projection [3]. The basic idea of LS is to evaluate the features according to their locality preserving power.

### 2.1 The Algorithm

Let $L_r$ denote the Laplacian Score of the $r$-th feature. Let $f_{ri}$ denote the $i$-th sample of the $r$-th feature, $i = 1, \cdots, m$. Our algorithm can be stated as follows:

1. Construct a nearest neighbor graph $G$ with $m$ nodes. The $i$-th node corresponds to $\mathbf{x}_i$. We put an edge between nodes $i$ and $j$ if $\mathbf{x}_i$ and $\mathbf{x}_j$ are "close", i.e. $\mathbf{x}_i$ is among $k$ nearest neighbors of $\mathbf{x}_j$ or $\mathbf{x}_j$ is among $k$ nearest neighbors of $\mathbf{x}_i$. When the label information is available, one can put an edge between two nodes sharing the same label.

2. If nodes $i$ and $j$ are connected, put $S_{ij} = e^{-\frac{\|\mathbf{x}_i - \mathbf{x}_j\|^2}{t}}$, where $t$ is a suitable constant. Otherwise, put $S_{ij} = 0$. The weight matrix $S$ of the graph models the local structure of the data space.

3. For the $r$-th feature, we define:
$$\mathbf{f}_r = [f_{r1}, f_{r2}, \cdots, f_{rm}]^T, D = diag(S\mathbf{1}), \mathbf{1} = [1, \cdots, 1]^T, L = D - S$$
where the matrix $L$ is often called graph Laplacian [2]. Let
$$\widetilde{\mathbf{f}}_r = \mathbf{f}_r - \frac{\mathbf{f}_r^T D \mathbf{1}}{\mathbf{1}^T D \mathbf{1}} \mathbf{1}$$

4. Compute the Laplacian Score of the $r$-th feature as follows:
$$L_r = \frac{\widetilde{\mathbf{f}}_r^T L \widetilde{\mathbf{f}}_r}{\widetilde{\mathbf{f}}_r^T D \widetilde{\mathbf{f}}_r} \tag{1}$$

## 3 Justification

### 3.1 Objective Function

Recall that given a data set we construct a weighted graph $G$ with edges connecting nearby points to each other. $S_{ij}$ evaluates the similarity between the $i$-th and $j$-th nodes. Thus,

the importance of a feature can be thought of as the degree it respects the graph structure. To be specific, a "good" feature should the one on which two data points are close to each other if and only if there is an edge between these two points. A reasonable criterion for choosing a good feature is to minimize the following object function:

$$L_r = \frac{\sum_{ij}(f_{ri} - f_{rj})^2 S_{ij}}{Var(\mathbf{f}_r)} \tag{2}$$

where $Var(\mathbf{f}_r)$ is the estimated variance of the $r$-th feature. By minimizing $\sum_{ij}(f_{ri} - f_{rj})^2 S_{ij}$, we prefer those features respecting the pre-defined graph structure. For a good feature, the bigger $S_{ij}$, the smaller $(f_{ri} - f_{rj})$, and thus the Laplacian Score tends to be small. Following some simple algebraic steps, we see that

$$\sum_{ij}(f_{ri} - f_{rj})^2 S_{ij} = \sum_{ij}\left(f_{ri}^2 + f_{rj}^2 - 2f_{ri}f_{rj}\right)S_{ij}$$

$$= 2\sum_{ij}f_{ri}^2 S_{ij} - 2\sum_{ij}f_{ri}S_{ij}f_{rj} = 2\mathbf{f}_r^T D\mathbf{f}_r - 2\mathbf{f}_r^T S\mathbf{f}_r = 2\mathbf{f}_r^T L\mathbf{f}_r$$

By maximizing $Var(\mathbf{f}_r)$, we prefer those features with large variance which have more representative power. Recall that the variance of a random variable $a$ can be written as follows:

$$Var(a) = \int_{\mathcal{M}}(a - \mu)^2 dP(a), \quad \mu = \int_{\mathcal{M}} a\, dP(a)$$

where $\mathcal{M}$ is the data manifold, $\mu$ is the expected value of $a$ and $dP$ is the probability measure. By spectral graph theory [2], $dP$ can be estimated by the diagonal matrix $D$ on the sample points. Thus, the *weighted* data variance can be estimated as follows:

$$Var(\mathbf{f}_r) = \sum_i (\mathbf{f}_{ri} - \mu_r)^2 D_{ii}$$

$$\mu_r = \sum_i \left(\mathbf{f}_{ri}\frac{D_{ii}}{\sum_i D_{ii}}\right) = \frac{1}{\left(\sum_i D_{ii}\right)}\left(\sum_i \mathbf{f}_{ri}D_{ii}\right) = \frac{\mathbf{f}_r^T D\mathbf{1}}{\mathbf{1}^T D\mathbf{1}}$$

To remove the mean from the samples, we define:

$$\widetilde{\mathbf{f}}_r = \mathbf{f}_r - \frac{\mathbf{f}_r^T D\mathbf{1}}{\mathbf{1}^T D\mathbf{1}}\mathbf{1}$$

Thus,

$$Var(\mathbf{f}_r) = \sum_i \widetilde{\mathbf{f}}_{ri}^2 D_{ii} = \widetilde{\mathbf{f}}_r^T D\widetilde{\mathbf{f}}_r$$

Also, it is easy to show that $\widetilde{\mathbf{f}}_r^T L\widetilde{\mathbf{f}}_r = \mathbf{f}_r^T L\mathbf{f}_r$ (please see Proposition 1 in Section 4.2 for detials). We finally get equation (1).

It would be important to note that, if we do not remove the mean, the vector $\mathbf{f}_r$ can be a non-zero constant vector such as $\mathbf{1}$. It is easy to check that, $\mathbf{1}^T L\mathbf{1} = 0$ and $\mathbf{1}^T D\mathbf{1} > 0$. Thus, $L_r = 0$. Unfortunately, this feature is clearly of no use since it contains no information. With mean being removed, the new vector $\widetilde{\mathbf{f}}_r$ is orthogonal to $\mathbf{1}$ with respect to $D$, i.e. $\widetilde{\mathbf{f}}_r^T D\mathbf{1} = 0$. Therefore, $\widetilde{\mathbf{f}}_r$ can not be any constant vector other than $\mathbf{0}$. If $\widetilde{\mathbf{f}}_r = \mathbf{0}, \widetilde{\mathbf{f}}_r^T L\widetilde{\mathbf{f}}_r = \widetilde{\mathbf{f}}_r^T D\widetilde{\mathbf{f}}_r = 0$. Thus, the Laplacian Score $L_r$ becomes a trivial solution and the $r$-th feature is excluded from selection. While computing the weighted variance, the matrix $D$ models the importance (or local density) of the data points. We can also simply replace it by the identity matrix $I$, in which case the weighted variance becomes the standard variance. To be specific,

$$\widetilde{\mathbf{f}}_r = \mathbf{f}_r - \frac{\mathbf{f}_r^T I\mathbf{1}}{\mathbf{1}^T I\mathbf{1}}\mathbf{1} = \mathbf{f}_r - \frac{\mathbf{f}_r^T \mathbf{1}}{n}\mathbf{1} = \mathbf{f}_r - \mu\mathbf{1}$$

where $\mu$ is the mean of $f_{ri}, i = 1, \cdots, n$. Thus,

$$Var(\mathbf{f}_r) = \widetilde{\mathbf{f}}_r^T I \widetilde{\mathbf{f}}_r = \frac{1}{n}(\mathbf{f}_r - \mu\mathbf{1})^T(\mathbf{f}_r - \mu\mathbf{1}) \tag{3}$$

which is just the *standard* variance.

In fact, the Laplacian scores can be thought of as the Rayleigh quotients for the features with respect to the graph $G$, please see [2] for details.

## 3.2 Connection to Fisher Score

In this section, we provide a theoretical analysis of the connection between our algorithm and the canonical Fisher score.

Given a set of data points with label, $\{\mathbf{x}_i, y_i\}_{i=1}^n$, $y_i \in \{1, \cdots, c\}$. Let $n_i$ denote the number of data points in class $i$. Let $\mu_i$ and $\sigma_i^2$ be the mean and variance of class $i$, $i = 1, \cdots, c$, corresponding to the $r$-$th$ feature. Let $\mu$ and $\sigma^2$ denote the mean and variance of the whole data set. The Fisher score is defined below:

$$F_r = \frac{\sum_{i=1}^c n_i(\mu_i - \mu)^2}{\sum_{i=1}^c n_i\sigma_i^2} \tag{4}$$

In the following, we show that Fisher score is equivalent to Laplacian score with a special graph structure. We define the weight matrix as follows:

$$S_{ij} = \begin{cases} \frac{1}{n_l}, & y_i = y_j = l; \\ 0, & \text{otherwise.} \end{cases} \tag{5}$$

Without loss of generality, we assume that the data points are ordered according to which class they are in, so that $\{\mathbf{x}_1, \cdots, \mathbf{x}_{n_1}\}$ are in the first class, $\{\mathbf{x}_{n_1+1}, \cdots, \mathbf{x}_{n_1+n_2}\}$ are in the second class, etc. Thus, $S$ can be written as follows:

$$S = \begin{pmatrix} S_1 & 0 & 0 \\ 0 & \ddots & 0 \\ 0 & 0 & S_c \end{pmatrix}$$

where $S_i = \frac{1}{n_i}\mathbf{1}\mathbf{1}^T$ is an $n_i \times n_i$ matrix. For each $S_i$, the raw (or column) sum is equal to 1, so $D_i = diag(S_i\mathbf{1})$ is just the identity matrix. Define $\mathbf{f}_r^1 = [f_{r1}, \cdots, f_{rn_1}]^T$, $\mathbf{f}_r^2 = [f_{r,n_1+1}, \cdots, f_{r,n_1+n_2}]^T$, etc. We now make the following observations.

**Observation 1** With the weight matrix $S$ defined in (5), we have $\widetilde{\mathbf{f}}_r^T L \widetilde{\mathbf{f}}_r = \mathbf{f}_r^T L \mathbf{f}_r = \sum_i n_i\sigma_i^2$, where $L = D - S$.

To see this, define $L_i = D_i - S_i = I_i - S_i$, where $I_i$ is the $n_i \times n_i$ identity matrix. We have

$$\mathbf{f}_r^T L \mathbf{f}_r = \sum_{i=1}^c (\mathbf{f}_r^i)^T L_i \mathbf{f}_r^i = \sum_{i=1}^c (\mathbf{f}_r^i)^T (I_i - \frac{1}{n_i}\mathbf{1}\mathbf{1}^T)\mathbf{f}_r^i = \sum_{i=1}^c n_i cov(\mathbf{f}_r^i, \mathbf{f}_r^i) = \sum_{i=1}^c n_i\sigma_i^2$$

Note that, since $\mathbf{u}^T L\mathbf{1} = \mathbf{1}^T L\mathbf{u} = 0$, $\forall \mathbf{u} \in \mathbf{R}^n$, the value of $\mathbf{f}_r^T L \mathbf{f}_r$ remains unchanged by subtracting a constant vector $(= \alpha\mathbf{1})$ from $\mathbf{f}_r$. This shows that $\widetilde{\mathbf{f}}_r^T L \widetilde{\mathbf{f}}_r = \mathbf{f}_r^T L \mathbf{f}_r = \sum_i n_i\sigma_i^2$.

**Observation 2** With the weight matrix $S$ defined in (5), we have $\widetilde{\mathbf{f}}_r^T D \widetilde{\mathbf{f}}_r = n\sigma^2$.

To see this, by the definition of $S$, we have $D = I$. Thus, this is a immediate result from equation (3).

**Observation 3** With the weight matrix $S$ defined in (5), we have $\sum_{i=1}^{c} n_i(\mu_i - \mu)^2 = \widetilde{\mathbf{f}}_r^T D\widetilde{\mathbf{f}}_r - \widetilde{\mathbf{f}}_r^T L\widetilde{\mathbf{f}}_r$.

To see this, notice

$$\sum_{i=1}^{c} n_i(\mu_i - \mu)^2 = \sum_{i=1}^{c} \left( n_i \mu_i^2 - 2n_i \mu_i \mu + n_i \mu^2 \right)$$

$$= \sum_{i=1}^{c} \frac{1}{n_i}(n_i \mu_i)^2 - 2\mu \sum_{i=1}^{c} n_i \mu_i + \mu^2 \sum_{i=1}^{c} n_i = \sum_{i=1}^{c} \frac{1}{n_i}\left( (\mathbf{f}_r^i)^T \mathbf{1}\mathbf{1}^T \mathbf{f}_r^i \right) - 2n\mu^2 + n\mu^2$$

$$= \sum_{i=1}^{c} \mathbf{f}_r^i S_i \mathbf{f}_r^i - \frac{1}{n}(n\mu)^2 = \mathbf{f}_r^T S\mathbf{f}_r - \mathbf{f}_r^T (\frac{1}{n}\mathbf{1}\mathbf{1}^T)\mathbf{f}_r$$

$$= \mathbf{f}_r^T(I - S)\mathbf{f}_r - \mathbf{f}_r^T(I - \frac{1}{n}\mathbf{1}\mathbf{1}^T)\mathbf{f}_r = \mathbf{f}_r^T L\mathbf{f}_r - n\sigma^2 = \widetilde{\mathbf{f}}_r^T L\widetilde{\mathbf{f}}_r - \widetilde{\mathbf{f}}_r^T D\widetilde{\mathbf{f}}_r$$

This completes the proof.

We therefore get the following relationship between the Laplacian score and Fisher score:

**Theorem 1** *Let $F_r$ denote the Fisher score of the $r$-th feature. With the weight matrix $S$ defined in (5), we have $L_r = \frac{1}{1+F_r}$.*

**Proof** From observations 1,2,3, we see that

$$F_r = \frac{\sum_{i=1}^{c} n_i(\mu_i - \mu)^2}{\sum_{i=1}^{c} n_i \sigma_i^2} = \frac{\widetilde{\mathbf{f}}_r^T D\widetilde{\mathbf{f}}_r - \widetilde{\mathbf{f}}_r^T L\widetilde{\mathbf{f}}_r}{\widetilde{\mathbf{f}}_r^T L\widetilde{\mathbf{f}}_r} = \frac{1}{L_r} - 1$$

Thus, $L_r = \frac{1}{1+F_r}$.  ∎

## 4  Experimental Results

Several experiments were carried out to demonstrate the efficiency and effectiveness of our algorithm. Our algorithm is a unsupervised filter method, while almost all the existing filter methods are supervised. Therefore, we compared our algorithm with data variance which can be performed in unsupervised fashion.

### 4.1  UCI Iris Data

Iris dataset, popularly used for testing clustering and classification algorithms, is taken from UCI ML repository. It contains 3 classes of 50 instances each, where each class refers to a type of Iris plant. Each instance is characterized by four features, i.e. sepal length, sepal width, petal length, and petal width. One class is linearly separable from the other two, but the other two are not linearly separable from each other. Out of the four features it is known that the features F3 (petal length) and F4 (petal width) are more important for the underlying clusters.

The class correlation for each feature is 0.7826, -0.4194, 0.9490 and 0.9565. We also used leave-one-out strategy to do classification by using each single feature. We simply used the nearest neighbor classifier. The classification error rates for the four features are 0.41, 0.52, 0.12 and 0.12, respectively. Our analysis indicates that F3 and F4 are better than F1 and F2 in the sense of discrimination. In figure 1, we present a 2-D visualization of the Iris data.

We compared three methods, i.e. Variance, Fisher score and Laplacian Score for feature selection. All of them are filter methods which are independent to any learning tasks. However, Fisher score is supervised, while the other two are unsupervised.

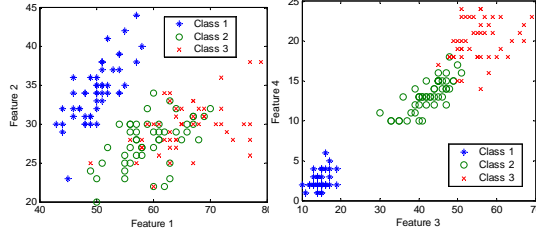

Figure 1: 2-D visualization of the Iris data.

By using variance, the four features are sorted as F3, F1, F4, F2. Laplacian score (with $k \geq 15$) sorts these four features as F3, F4, F1, F2. Laplacian score (with $3 \leq k < 15$) sorts these four features as F4, F3, F1, F2. With a larger $k$, we see more global structure of the data set. Therefore, the feature F3 is ranked above F4 since the variance of F3 is greater than that of F4. By using Fisher score, the four features are sorted as F3, F4, F1, F2. This indicates that Laplacian score (unsupervised) achieved the same result as Fisher score (supervised).

## 4.2 Face Clustering on PIE

In this section, we apply our feature selection algorithm to face clustering. By using Laplacian score, we select a subset of features which are the most useful for discrimination. Clustering is then performed in such a subspace.

### 4.2.1 Data Preparation

The CMU PIE face database is used in this experiment. It contains 68 subjects with 41,368 face images as a whole. Preprocessing to locate the faces was applied. Original images were normalized (in scale and orientation) such that the two eyes were aligned at the same position. Then, the facial areas were cropped into the final images for matching. The size of each cropped image is $32 \times 32$ pixels, with 256 grey levels per pixel. Thus, each image is represented by a 1024-dimensional vector. No further preprocessing is done. In this experiment, we fixed the pose and expression. Thus, for each subject, we got 24 images under different lighting conditions.

For each given number $k$, $k$ classes were randomly selected from the face database. This process was repeated 20 times (except for $k = 68$) and the average performance was computed. For each test (given $k$ classes), two algorithms, i.e. feature selection using variance and Laplacian score are used to select the features. The K-means was then performed in the selected feature subspace. Again, the K-means was repeated 10 times with different initializations and the best result in terms of the objective function of K-means was recorded.

### 4.2.2 Evaluation Metrics

The clustering result is evaluated by comparing the obtained label of each data point with that provided by the data corpus. Two metrics, the accuracy ($AC$) and the normalized mutual information metric ($\overline{MI}$) are used to measure the clustering performance [6]. Given a data point $\mathbf{x}_i$, let $r_i$ and $s_i$ be the obtained cluster label and the label provided by the data corpus, respectively. The $AC$ is defined as follows:

$$AC = \frac{\sum_{i=1}^{n} \delta(s_i, map(r_i))}{n} \tag{6}$$

where $n$ is the total number of data points and $\delta(x, y)$ is the delta function that equals one if $x = y$ and equals zero otherwise, and map($r_i$) is the permutation mapping function that

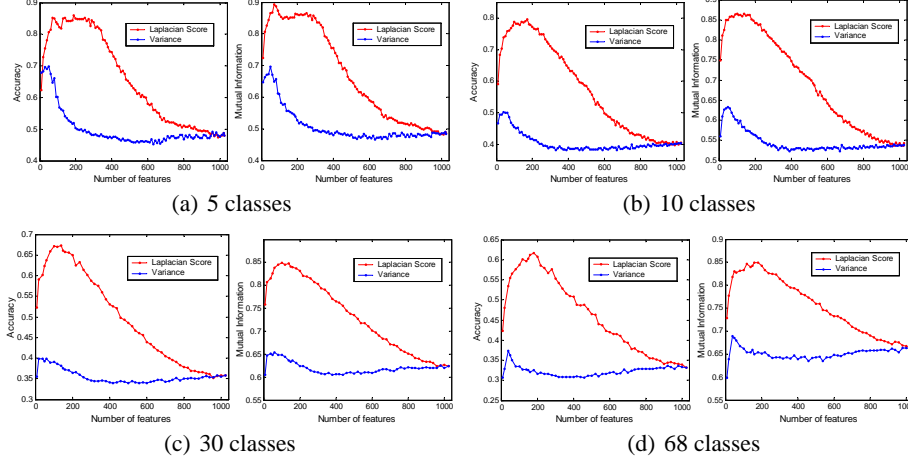

(a) 5 classes                      (b) 10 classes

(c) 30 classes                    (d) 68 classes

Figure 2: Clustering performance versus number of features

maps each cluster label $r_i$ to the equivalent label from the data corpus. The best mapping can be found by using the Kuhn-Munkres algorithm [5].

Let $C$ denote the set of clusters obtained from the ground truth and $C'$ obtained from our algorithm. Their mutual information metric $MI(C, C')$ is defined as follows:

$$MI(C, C') = \sum_{c_i \in C, c'_j \in C'} p(c_i, c'_j) \cdot log_2 \frac{p(c_i, c'_j)}{p(c_i) \cdot p(c'_j)} \tag{7}$$

where $p(c_i)$ and $p(c'_j)$ are the probabilities that a data point arbitrarily selected from the corpus belongs to the clusters $c_i$ and $c'_j$, respectively, and $p(c_i, c'_j)$ is the joint probability that the arbitrarily selected data point belongs to the clusters $c_i$ as well as $c'_j$ at the same time. In our experiments, we use the normalized mutual information $\overline{MI}$ as follows:

$$\overline{MI}(C, C') = \frac{MI(C, C')}{\max(H(C), H(C'))} \tag{8}$$

where $H(C)$ and $H(C')$ are the entropies of $C$ and $C'$, respectively. It is easy to check that $\overline{MI}(C, C')$ ranges from 0 to 1. $\overline{MI} = 1$ if the two sets of clusters are identical, and $\overline{MI} = 0$ if the two sets are independent.

### 4.2.3 Results

We compared Laplacian score with data variance for clustering. Note that, we did not compare with Fisher score because it is supervised and the label information is not available in the clustering experiments. Several tests were performed with different numbers of clusters (k=5, 10, 30, 68). In all the tests, the number of nearest neighbors in our algorithm is taken to be 5. The experimental results are shown in Figures 2 and Table 1. As can be seen, in all these cases, our algorithm performs much better than using variance for feature selection. The clustering performance varies with the number of features. The best performance is obtained at very low dimensionality (less than 200). This indicates that feature selection is capable of enhancing clustering performance. In Figure 3, we show the selected features in the image domain for each test (k=5, 10, 30, 68), using our algorithm, data variance and Fisher score. The brightness of the pixels indicates their importance. That is, the more bright the pixel is, the more important. As can be seen, Laplacian score provides better approximation to Fisher score than data variance. Both Laplacian score

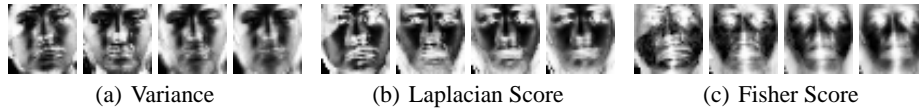

| (a) Variance | (b) Laplacian Score | (c) Fisher Score |

Figure 3: Selected features in the image domain, $k = 5, 10, 30, 68$. The brightness of the pixels indicates their importance.

Table 1: Clustering performance comparisons ($k$ is the number of clusters)

| | | Accuracy | | | | | | |
|---|---|---|---|---|---|---|---|---|
| k | Feature Number | 20 | 50 | 100 | 200 | 300 | 500 | 1024 |
| 5 | Laplacian Score | 0.727 | 0.806 | 0.831 | 0.849 | 0.837 | 0.644 | 0.479 |
| | Variance | 0.683 | 0.698 | 0.602 | 0.503 | 0.482 | 0.464 | 0.479 |
| 10 | Laplacian Score | 0.685 | 0.743 | 0.787 | 0.772 | 0.711 | 0.585 | 0.403 |
| | Variance | 0.494 | 0.500 | 0.456 | 0.418 | 0.392 | 0.392 | 0.403 |
| 30 | Laplacian Score | 0.591 | 0.623 | 0.671 | 0.650 | 0.588 | 0.485 | 0.358 |
| | Variance | 0.399 | 0.393 | 0.390 | 0.365 | 0.346 | 0.340 | 0.358 |
| 68 | Laplacian Score | 0.479 | 0.554 | 0.587 | 0.608 | 0.553 | 0.465 | 0.332 |
| | Variance | 0.328 | 0.362 | 0.334 | 0.316 | 0.311 | 0.312 | 0.332 |
| | | Mutual Information | | | | | | |
| k | Feature Number | 20 | 50 | 100 | 200 | 300 | 500 | 1024 |
| 5 | Laplacian Score | 0.807 | 0.866 | 0.861 | 0.862 | 0.85 | 0.652 | 0.484 |
| | Variance | 0.662 | 0.697 | 0.609 | 0.526 | 0.495 | 0.482 | 0.484 |
| 10 | Laplacian Score | 0.811 | 0.849 | 0.865 | 0.842 | 0.796 | 0.705 | 0.538 |
| | Variance | 0.609 | 0.632 | 0.6 | 0.563 | 0.538 | 0.529 | 0.538 |
| 30 | Laplacian Score | 0.807 | 0.826 | 0.849 | 0.831 | 0.803 | 0.735 | 0.624 |
| | Variance | 0.646 | 0.649 | 0.649 | 0.624 | 0.611 | 0.608 | 0.624 |
| 68 | Laplacian Score | 0.778 | 0.83 | 0.833 | 0.843 | 0.814 | 0.76 | 0.662 |
| | Variance | 0.639 | 0.686 | 0.661 | 0.651 | 0.642 | 0.643 | 0.662 |

and Fisher score have the brightest pixels in the area of two eyes, nose, mouth, and face contour. This indicates that even though our algorithm is unsupervised, it can discover the most discriminative features to some extent.

## 5 Conclusions

In this paper, we propose a new filter method for feature selection which is independent to any learning tasks. It can be performed in either supervised or unsupervised fashion. The new algorithm is based on the observation that local geometric structure is crucial for discrimination. Experiments on Iris data set and PIE face data set demonstrate the effectiveness of our algorithm.

## References

[1] M. Belkin and P. Niyogi, "Laplacian Eigenmaps and Spectral Techniques for Embedding and Clustering," *Advances in Neural Information Processing Systems*, Vol. 14, 2001.

[2] Fan R. K. Chung, *Spectral Graph Theory,* Regional Conference Series in Mathematics, number 92, 1997.

[3] X. He and P. Niyogi, "Locality Preserving Projections," *Advances in Neural Information Processing Systems*, Vol. 16, 2003.

[4] R. Kohavi and G. John, "Wrappers for Feature Subset Selection," *Artificial Intelligence*, 97(1-2):273-324, 1997.

[5] L. Lovasz and M. Plummer, *Matching Theory*, Akadémiai Kiadó, North Holland, 1986.

[6] W. Xu, X. Liu and Y. Gong, "Document Clustering Based on Non-negative Matrix Factorization," *ACM SIGIR Conference on Information Retrieval*, 2003.
